# An Infinite Factor Model Hierarchy
# Via a Noisy-Or Mechanism

**Aaron C. Courville, Douglas Eck and Yoshua Bengio**
Department of Computer Science and Operations Research
University of Montréal
Montréal, Québec, Canada
{courvila,eckdoug,bengioy}@iro.umontreal.ca

## Abstract

The Indian Buffet Process is a Bayesian nonparametric approach that models objects as arising from an infinite number of latent factors. Here we extend the latent factor model framework to two or more unbounded layers of latent factors. From a generative perspective, each layer defines a conditional *factorial* prior distribution over the binary latent variables of the layer below via a noisy-or mechanism. We explore the properties of the model with two empirical studies, one digit recognition task and one music tag data experiment.

## 1 Introduction

The Indian Buffet Process (IBP) [5] is a Bayesian nonparametric approach that models objects as arising from an unbounded number of latent features. One of the main motivations for the IBP is the desire for a *factorial* representation of data, with each element of the data vector modelled independently, i.e. as a collection of factors rather than as monolithic wholes as assumed by other modeling paradigms such as mixture models. Consider music tag data collected through the internet service provider Last.fm. Users of the service label songs and artists with descriptive tags that collectively form a representation of an artist or song. These tags can then be used to organize playlists around certain themes, such as *music from the 80's*. The top 8 tags for the popular band RADIOHEAD are: *alternative*, *rock*, *alternative rock*, *indie*, *electronic*, *britpop*, *british*, and *indie rock*. The tags point to various facets of the band, for example that they are based in Britain, that they make use of electronic music and that their style of music is alternative and/or rock. These facets or features are not mutually exclusive properties but represent some set of distinct aspects of the band.

Modeling such data with an IBP allows us to capture the latent factors that give rise to the tags, including inferring the number of factors characterizing the data. However the IBP assumes these latent features are independent across object instances. Yet in many situations, a more compact and/or accurate description of the data could be obtained if we were prepared to consider *dependencies between latent factors*. Despite there being a wealth of distinct factors that collectively describe an artist, it is clear that the co-occurrence of some features is more likely than others. For example, factors associated with the tag *alternative* are more likely to co-occur with those associated with the tag *indie* than those associated with tag *classical*.

The main contribution of this work is to present a method for extending infinite latent factor models to two or more unbounded layers of factors, with upper-layer factors defining a *factorial* prior distribution over the binary factors of the layer below. In this framework, the upper-layer factors express correlations between lower-layer factors via a noisy-or mechanism. Thus our model may be interpreted as a Bayesian nonparametric version of the noisy-or network [6, 8]. In specifying the model and inference scheme, we make use of the recent stick-breaking construction of the IBP [10].

For simplicity of presentation, we focus on a two-layer hierarchy, though the method extends readily to higher-order cases. We show how the complete model is amenable to efficient inference via a Gibbs sampling procedure and compare performance of our hierarchical method with the standard IBP construction on both a digit modeling task, and a music genre-tagging task.

## 2   Latent Factor Modeling

Consider a set of $N$ objects or exemplars: $x_{1:N} = [x_1, x_2, \ldots, x_N]$. We model the $n$th object with the distribution $x_n \mid z_{n,1:K}, \theta \sim F(z_{n,1:K}, \theta_{1:K})$, with model parameters $\theta_{1:K} = [\theta_k]_{k=1}^K$ (where $\theta_k \sim H$ indep. $\forall k$) and feature variables $z_{n,1:K} = [z_{nk}]_{k=1}^K$ which we take to be binary: $z_{nk} \in \{0, 1\}$. We denote the presence of feature $k$ in example $n$ as $z_{nk} = 1$ and its absence as $z_{nk} = 0$. Features present in an object are said to be *active* while absent features are *inactive*. Collectively, the features form a typically sparse binary $N \times K$ feature matrix, which we denote as $z_{1:N,1:K}$, or simply $Z$. For each feature $k$ let $\mu_k$ be the prior probability that the feature is active. The collection of $K$ probabilities: $\mu_{1:K}$, are assumed to be mutually independent, and distributed according to a $\text{Beta}(\alpha/\mathrm{K}, 1)$ prior. Summarizing the full model, we have (indep.$\forall n, k$):

$$x_n \mid z_{n,1:K}, \theta \sim F(z_{n,1:K}, \theta) \qquad z_{nk} \mid \mu_k \sim \text{Bernoulli}(\mu_k) \qquad \mu_k \mid \alpha \sim \text{Beta}\left(\frac{\alpha}{K}, 1\right)$$

According to the standard development of the IBP, we can marginalize over variables $\mu_{1:K}$ and take the limit $K \to \infty$ to recover a distribution over an unbounded binary feature matrix $Z$. In the development of the inference scheme for our hierarchical model, we make use of an alternative characterization of the IBP: the IBP stick-breaking construction [10]. As with the stick-breaking construction of the Dirichlet process (DP), the IBP stick-breaking construction provides a direct characterization of the random latent feature probabilities via an unbounded sequence. Consider once again the finite latent factor model described above. Letting $K \to \infty$, $Z$ now possesses an unbounded number of columns with a corresponding unbounded set of random probabilities $[\mu_1, \mu_2, \ldots]$. Re-arranged in decreasing order: $\mu_{(1)} > \mu_{(2)} > \ldots$, these factor probabilities can be expressed recursively as: $\mu_{(k)} = U_{(k)}\mu_{(k-1)} = \prod_{(l)} U_{(l)}$, where $U_{(k)} \overset{i.i.d}{\sim} \text{Beta}(\alpha, 1)$.

## 3   A Hierarchy of Latent Features Via a Noisy-OR Mechanism

In this section we extend the infinite latent features framework to incorporate interactions between multiple layers of unbounded features. We begin by defining a finite version of the model before considering the limiting process. We consider here the simplest hierarchical latent factor model consisting of two layers of binary latent features: an upper-layer binary latent feature matrix $Y$ with elements $y_{nj}$, and a lower-layer binary latent feature matrix $Z$ with elements $z_{nk}$. The probability distribution over the elements $y_{nj}$ is defined as previously in the limit construction of the IBP: $y_{nj} \mid \mu_j \sim \text{Bernoulli}(\mu_j)$, with $\mu_j \mid \alpha_\mu \sim \text{Beta}(\alpha_\mu/J, 1)$. The lower binary variables $z_{nk}$ are also defined as Bernoulli distributed random quantities:

$$z_{nk} \mid y_{n,:}, V_{:,k} \sim \text{Bernoulli}(1 - \prod_j (1 - y_{nj}V_{jk})) \qquad \text{indep.}\forall n, k. \qquad (1)$$

However, here the probability that $z_{nk} = 1$ is a function of the upper binary variables $y_{n,:}$ and the $k$th column of the weight matrix $V$, with probabilities $V_{jk} \in [0, 1]$ connecting $y_{nj}$ to $z_{nk}$. The crux of the model is how $y_{nj}$ interacts with $z_{nk}$ via a *noisy-or* mechanism defined in Eq. (1). The binary $y_{nj}$ modulates the involvement of the $V_{jk}$ terms in the product, which in turn modulates $P(z_{nk} = 1 \mid y_{n,:}, V_{:,k})$. The noisy-or mechanism interacts positively in the sense that changing an element $y_{nj}$ from inactive to active can only increase $P(z_{nk} = 1 \mid y_{n,:}, V_{:k})$, or leave it unchanged in the case where $V_{jk} = 0$. We interpret the active $y_{n,:}$ to be possible *causes* of the activation of the individual $z_{nk}, \forall k$. Through the weight matrix $V$, every element of $Y_{n,1:J}$ is connected to every element of $Z_{n,1:K}$, thus $V$ is a random matrix of size $J \times K$. In the case of finite $J$ and $K$, an obvious choice of prior for $V$ is: $V_{jk} \overset{i.i.d}{\sim} \text{Beta}(a, b)$, $\forall j, k$. However, looking ahead to the case where $J \to \infty$ and $K \to \infty$, the prior over $V$ will require some additional structure.

Recently, [11] introduced the Hierarchical Beta Process (HBP) and elucidated the relationship between this and the Indian Buffet Process. We use a variant of the HBP to define a prior over $V$:

$$\nu_k \sim \text{Beta}(\alpha_\nu/K, 1) \qquad V_{jk} \mid \nu_k \sim \text{Beta}(c\nu_k, c(1 - \nu_k) + 1) \qquad \text{indep.}\forall k, j, \qquad (2)$$

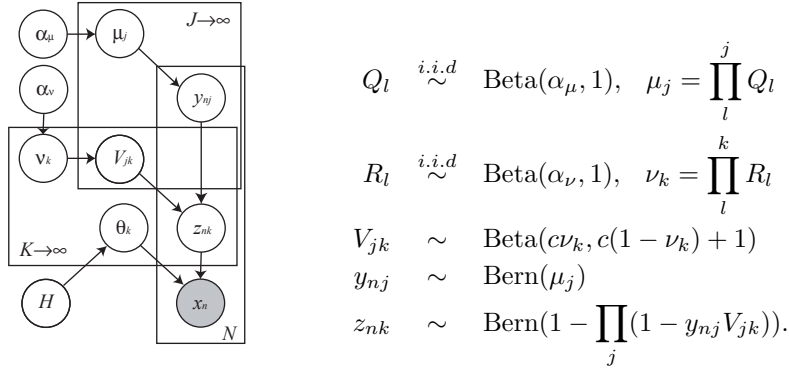

$$Q_l \overset{i.i.d}{\sim} \text{Beta}(\alpha_\mu, 1), \quad \mu_j = \prod_l^j Q_l$$

$$R_l \overset{i.i.d}{\sim} \text{Beta}(\alpha_\nu, 1), \quad \nu_k = \prod_l^k R_l$$

$$V_{jk} \sim \text{Beta}(c\nu_k, c(1 - \nu_k) + 1)$$
$$y_{nj} \sim \text{Bern}(\mu_j)$$
$$z_{nk} \sim \text{Bern}(1 - \prod_j (1 - y_{nj}V_{jk})).$$

Figure 1: Left: A graphical representation of the 2-layer hierarchy of infinite binary factor models. Right: Summary of the hierarchical infinite noisy-or factor model in the stick-breaking parametrization.

where each column of $V$ (indexed by $k$) is constrained to share a common prior. Structuring the prior this way allows us to maintain a well behaved prior over the $Z$ matrix as we let $K \to \infty$, grouping the values of $V_{jk}$ across $j$ while $\mathbb{E}[\nu_k] \to 0$. However beyond the region of very small $\nu_k$ ($0 < \nu_k << 1$), we would like the weights $V_{jk}$ to vary more independently. Thus we modify the model of [11] to include the $+1$ term to the prior over $V_{jk}$ (in Eq. (2)) and we limit $c \leq 1$. Fig. 1 shows a graphical representation of the complete 2-layer hierarchical noisy-or factor model, as $J \to \infty$ and $K \to \infty$.

Finally, we augment the model with an additional random matrix $A$ with multinomial elements $A_{nk}$, assigning each instance of $z_{nk} = 1$ to an index $j$ corresponding to the active upper-layer unit $y_{nj}$ responsible for *causing* the event. The probability that $A_{nk} = j$ is defined via a familiar stick-breaking scheme. By enforcing an (arbitrary) ordering over the indices $j = [1, J]$, we can view the noisy-or mechanism defined in Eq. (1) as specifying, for each $z_{nk}$, an ordered series of binary trials (i.e. coin flips). For each $z_{nk}$, we proceed through the ordered set of elements, $\{V_{jk}, y_{nj}\}_{j=1,2,...}$, performing random trials. With probability $y_{n,j^*}V_{j^*,k}$, trial $j^*$ is deemed a "success" and we set $z_{nk} = 1$, $A_{nk} = j^*$, and no further trials are conducted for $\{n, k, j > j^*\}$. Conversely, with probability $(1 - y_{nj^*}V_{j^*k})$ the trial is deemed a "failure" and we move on to trial $j^* + 1$. Since all trials $j$ associated with inactive upper-layer features are failures with probability one (because $y_{nj} = 0$), we need only consider the trials for which $y_{nj} = 1$. If, for a given $z_{nk}$, all trials $j$ for which $y_{nj} = 1$ (active) are failures, then we set $z_{nk} = 0$ with probability one. The probability associated with the event $z_{nk} = 0$ is therefore given by the product of the failure probabilities for each of the $J$ trials: $P(z_{nk} = 0 \mid y_{n,:}, V_{:,k}) = \prod_{j=1}^{J}(1 - y_{nj}V_{jk})$, and with $P(z_{nk} = 1 \mid y_{n,:}, V_{:,k}) = 1 - P(z_{nk} = 0 \mid y_{n,:}, V_{:,k})$, we arrive at the noisy-or mechanism given in Eq. (1). This process is similar to the sampling process associated with the Dirichlet process stick-breaking construction [7]. Indeed, the process described above specifies a stick-breaking construction of a *generalized Dirichlet distribution* [1] over the multinomial probabilities corresponding to the $A_{nk}$. The generalized Dirichlet distribution defined in this way has the important property that it is conjugate to multinomial sampling.

With the generative process specified as above, we can define the posterior distribution over the weights $V$ given the assignment matrix $A$ and the latent feature matrix $Y$. Let $M_{jk} = \sum_{n=1}^{N} \mathbb{I}(A_{nk} = j)$ be the number of times that the $j$th trial was a success for $z_{:,k}$ (i.e. the number of times $y_{nj}$ *caused* the activation of $z_{nk}$) and let $N_{jk} = \sum_{n=1}^{N} y_{nj}\mathbb{I}(A_{nk} > j)$, that is the number of times that the $j$-th trial was a failure for $z_{nk}$ despite $y_{nj}$ being active. Finally, let us also denote the number of times $y_{:,j}$ is active: $N_j = \sum_{n=1}^{N} y_{nj}$. Given these quantities, the posterior distributions for the model parameters $\mu_j$ and $V_{jk}$ are given by:

$$\mu_j \mid Y \sim \text{Beta}(\alpha_\mu/J + N_j, 1 + N - N_j) \quad (3)$$
$$V_{jk} \mid Y, A \sim \text{Beta}(c\nu_k + M_{jk}, c(1 - \nu_k) + N_{jk} + 1) \quad (4)$$

These conjugate relationships are exploited in the Gibbs sampling procedure described in Sect. 4. By integrating out $V_{jk}$, we can recover (up to a constant) the posterior distribution over $\nu_k$:

$$p(\nu_k \mid A_{:,k}) \propto \nu_k^{\alpha_\nu/K-1} \prod_{j=1}^{J} \frac{\Gamma(c\nu_k + M_{jk})}{\Gamma(c\nu_k)} \frac{\Gamma(c(1-\nu_k) + N_{jk} + 1)}{\Gamma(c(1-\nu_k) + 1)} \tag{5}$$

One property of the marginal likelihood is that wholly inactive elements of $Y$, which we denote as $y_{:,j'} = 0$, do not impact the likelihood as $N_{j',k} = 0$, $M_{j',k} = 0$. This becomes particularly important as we let $J \to \infty$.

Having defined the finite model, it remains to take the limit as both $K \to \infty$ and $J \to \infty$. Taking the limit of $J \to \infty$ is relatively straightforward as the upper-layer factor model naturally tends to an IBP: $Y \sim$ IBP, and its involvement in the remainder of the model is limited to the set of active elements of $Y$, which remains finite for finite datasets. In taking $K \to \infty$, the distribution over the unbounded $\nu_k$ converges to that of the IBP, while the conditional distribution over the noisy-or weights $V_{jk}$ remain simple beta distributions given the corresponding $\nu_k$ (as in Eq. (4)).

## 4 Inference

In this section, we describe an inference strategy to draw samples from the model posterior. The algorithm is based jointly on the blocked Gibbs sampling strategy for truncated Dirichlet distributions [7] and on the IBP semi-ordered slice sampler [10], which we employ at each layer of the hierarchy. Because both algorithms are based on the strategy of directly sampling an instantiation of the model parameters, their use together permits us to define an efficient extended blocked Gibbs sampler over the entire model without approximation.

To facilitate our description of the semi-ordered slice sampler, we separate $\mu_{1:\infty}$ into two subsets: $\mu_{1:J^+}^+$ and $\mu_{1:\infty}^o$, where $\mu_{1:J^+}^+$ are the probabilities associated with the set of $J^+$ active upper-layer factors $Y^+$ (those that appear at least once in the dataset, i.e. $\exists i : y_{ij'}^+ = 1, 1 \leq j' \leq J^+$) and $\mu_{1:\infty}^o$ are associated with the unbounded set of inactive features $Y^o$ (those not appearing in the dataset). Similarly, we separate $\nu_{1:\infty}$ into $\nu_{1:K^+}^+$ and $\nu_{1:\infty}^o$, and $Z$ into corresponding active $Z^+$ and inactive $Z^o$ where $K^+$ is the number of active lower-layer factors.

### 4.1 Semi-ordered slice sampling of the upper-layer IBP

The IBP semi-ordered slice sampler maintains an unordered set of active $y_{1:N,1:J^+}^+$ with corresponding $\mu_{1:J^+}^+$ and $V_{1:J^+,1:K}$, while exploiting the IBP stick-breaking construction to sample from the distribution of ordered inactive features, up to an adaptively chosen truncation level controlled by an auxiliary slice variable $s_y$.

**Sample $s_y$.** The uniformly distributed auxiliary slice variables, $s_y$ controls the truncation level of the upper-layer IBP, where $\mu^*$ is defined as the smallest probability $\mu$ corresponding to an active feature:

$$s_y \mid Y, \mu_{1:\infty} \sim \text{Uniform}(0, \mu^*), \qquad \mu^* = \min\left\{1, \min_{1 \leq j' \leq J^+} \mu_{j'}^+\right\}. \tag{6}$$

As discussed in [10], the joint distribution is given by $p(s_y, \mu_{1:\infty}, Y) = p(Y, \mu_{1:\infty}) \times p(s_y \mid Y, \mu_{1:\infty})$, where marginalizing over $s_y$ preserves the original distribution over $Y$ and $\mu_{1:\infty}$. However, given $s_y$, the conditional distribution $p(y_{nj'} = 1 \mid Z, s_y, \mu_{1:\infty}) = 0$ for all $n, j'$ such that $\mu_{j'} < s_y$. This is the crux of the slice sampling approach: Each sample $s_y$ adaptively truncates the model, with $\mu_{1:J} > s_y$. Yet by marginalizing over $s_y$, we can recover samples from the original non-truncated distribution $p(Y, \mu_{1:\infty})$ without approximation.

**Sample $\mu_{1:J^o}^o$.** For the inactive features, we use adaptive rejection sampling (ARS) [4] to sequentially draw an ordered set of $J^o$ posterior feature probabilities from the distribution:

$$p(\mu_j^o \mid \mu_{j-1}^o, y_{:,\geq j}^o = 0) \propto \exp\left(\alpha_\mu \sum_{n=1}^{N} \frac{1}{n}(1-\mu_j^o)^n\right) \cdot (\mu_j^o)^{\alpha_\mu - 1}(1-\mu_j^o)^N \mathbb{I}(0 \leq \mu_j^o \leq \mu_{j-1}^o),$$

until $\mu_{J^o+1}^o < s_y$. The above expression arises from using the IBP stick-breaking construction to marginalize over the inactive elements of $\mu_:$ [10]. For each of the $J^o$ inactive features drawn, the

corresponding features $y_{1:N,1:J^o}^o$ are initialized to zero and the corresponding weight $V_{1:J^o,1:K}^o$ are sampled from their prior in Eq. (2). With the probabilities for both the active and a truncated set of inactive features sampled, the set of features are re-integrated into a set of $J = J^+ + J^o$ features $Y = [y_{1:N,1:J^+}^+, y_{1:N,1:J^o}^o]$ with probabilities $\mu_{1:J} = [\mu_{1:J^+}^+, \mu_{1:J^o}^o]$, and corresponding weights $V^T = [(V_{1:J^+,1:K}^+)^T, (V_{1:J^o,1:K}^o)^T]$.

**Sample $Y$.**  Given the upper-layer feature probabilities $\mu_{1:J}$, weight matrix $V$, and the lower-layer binary feature values $z_{nk}$, we update each $y_{nj}$ as follows:

$$p(y_{nj} = 1 \mid \mu_j, z_{n,:}, \mu^*) \propto \frac{\mu_j}{\mu^*} \prod_{k=1}^{K} p(z_{nk} \mid y_{nj} = 1, y_{n,\neg j}, V_{:,k}) \tag{7}$$

The denominator $\mu^*$ is subject to change if changing $y_{nj}$ induces a change in $\mu^*$ (as defined in Eq. (6)); $y_{n,\neg j}$ represents all elements $y_{n,1:J}$ except $y_{nj}$ The conditional probability of the lower-layer binary variables is given by: $p(z_{nk} \mid y_{n,:}, V_{:,k}) = (1 - \prod_j (1 - y_{nj} V_{jk}))$.

**Sample $\mu_{1:J^+}^+$.**  Once again we separate $Y$ and $\mu_{1:\infty}$ into a set of active features: $Y^+$ with probabilities $\mu_{1:J^+}^+$; and a set of inactive features $Y^o$ with $\mu_{1:\infty}^o$. The inactive set is discarded while the active set of $\mu_{1:J^+}^+$ are resampled from the posterior distribution: $\mu_j^+ \mid y_{:,j}^+ \sim \text{Beta}(N_j, 1 + N - N_j)$. At this point we also separate the lower-layer factors into an active set of $K+$ factors $Z^+$ with corresponding $\nu_{1:K^+}^+$, $V_{1:J^+,1:K^+}^+$ and data likelihood parameters $\theta^+$; and a discarded inactive set.

## 4.2   Semi-ordered slice sampling of the lower-layer factor model

Sampling the variables of the lower-layer IFM model proceeds analogously to the upper-layer IBP. However the presence of the hierarchical relationship between the $\nu_k$ and the $V_{:,k}$ (as defined in Eqs. (3) and (4)) does require some additional attention. We proceed by making use of the marginal distribution over the assignment probabilities to define a second auxiliary slice variable, $s_z$.

**Sample $s_z$.**  The auxiliary slice variable is sampled according to the following, where $\nu^*$ is defined as the smallest probability corresponding to an active feature:

$$s_z \mid Z, \nu_{1:\infty} \sim \text{Uniform}(0, \nu^*), \qquad \nu^* = \min\left\{1, \min_{1 \le k' \le K^+} \nu_{k'}^+\right\}.$$

**Sample $\nu_{1:K^o}^o$.**  Given $s_z$ and $Y$, the random probabilities over the *inactive* lower-layer binary features, $\nu_{1:\infty}^o$, are sampled sequentially to draw a set of $K^o$ feature probabilities, until $\nu_{K^o+1} < s_z$. The samples are drawn according to the distribution:

$$p(\nu_k^o \mid \nu_{k-1}^o, Y^+, z_{:,\ge k} = 0) \quad \propto \quad \mathbb{I}(0 \le \nu_k^o \le \nu_{k-1}^o)(\nu_k^o)^{\alpha_\nu - 1} \left(\prod_{j=1}^{J} \frac{\Gamma(c(1-\nu_k^o) + N_j)}{\Gamma(c(1-\nu_k^o))}\right) \times$$

$$\exp\left(\alpha_\nu \prod_{j=1}^{J} \frac{\Gamma(c)}{\Gamma(c + N_j)} \sum_{i=0}^{N_1 + \cdots + N_J} w_i c^i \sum_{l=1}^{i} \frac{1}{l}(1-\nu_k^o)^l\right). \tag{8}$$

Eq. (8) arises from the stick-breaking construction of the IBP and from the expression for $P(z_{:,>k}^o = 0 \mid \nu_k^o, Y^+)$ derived in the supplementary material [2]. Here we simply note that the $w_i$ are weights derived from the expansion of a product of terms involving unsigned Stirling numbers of the first kind. The distribution over the ordered inactive features is log-concave in $\log \nu_k$, and is therefore amenable to efficient sample via adaptive rejection sampling (as was done in sampling $\mu_{1:J^o}^o$). Each of the $K^o$ inactive features are initialized to zero for every data object, $Z^o = 0$, while the corresponding $V^o$ and likelihood parameters $\theta^o$ are drawn from their priors. Once the $\nu_{1:K^o}$ are drawn, both the active and inactive features of the lower-layer are re-integrated into the set of $K = K^+ + K^o$ features $Z = [Z^+, Z^o]$ with probabilities $\nu_{1:K} = [\nu_{1:K^+}^+, \nu_{1:K^o}^o]$ and corresponding weight matrix $V = [V_{1:J^+,1:K^+}^+, V_{1:J^+,1:K^o}^o]$ and parameters $\theta = [\theta^+, \theta^o]$.

**Sample $Z$.** Given $Y^+$ and $V$ we use Eq. (1) to specify the prior over $z_{1:N,1:K^*}$. Then, conditional on this prior, the data $X$ and parameters $\theta$, we sample sequentially for each $z_{nk}$:

$$p(z_{nk} \mid y_{n,:}^+, V_{:,k}, z_{n,\neg k}, \theta, \nu^*) = \frac{1}{\nu^*}\left(1 - \prod_{j=1}^{J^+}(1 - y_{nj}^+ V_{jk})\right) f(x_n \mid z_{n,:}, \theta),$$

where $f(x_n \mid z_{n,:}, \theta)$ is the likelihood function for the $n$th data object.

**Sample $A$.** Given $z_{nk}$, $y_{n,:}^+$ and $V_{:,k}$, we draw the multinomial variable $A_{nk}$ to assign responsibility, in the event $z_{ik} = 1$, to one of the upper-layer features $y_{nj}^+$,

$$p(A_{nk} = j \mid z_{nk} = 1, y_{n,:}^+, V_{:,k}) = V_{jk}\left[\prod_{i=1}^{j-1}(1 - y_{ni}^+ V_{ik})\right], \tag{9}$$

and if $y_{n,j'}^+ = 0$, $\forall j' > j^\dagger$, then $p(A_{nk} = j^\dagger \mid z_{nk} = 1, y_{n,:}^+, V_{:,k}) = \prod_{i=1}^{j^\dagger-1}(1 - y_{ni}^+ V_{ik})$ to ensure normalization of the distribution. If $z_{nk} = 0$, then $P(A_{nk} = \infty) = 1$.

**Sample $V$ and $\nu_{1:K^+}^+$.** Conditional on $Y^+$, $Z$ and $A$, the weights $V$ are resampled from Eq. (4), following the blocked Gibbs sampling procedure of [7]. Given the assignments $A$, the posterior of $\nu_k^+$ is given (up to a constant) by Eq. (5). This distribution is log concave in $\nu_k^+$, therefore we can once again use ARS to draw samples of the posterior of $\nu_k^+$, $1 \leq k \leq K^+$.

## 5  Experiments

In this section, we present two experiments to highlight the properties and capabilities of our hierarchical infinite factor model. Our goal is to assess, in these two cases, the impact of including an additional modeling layer. To this end, and in each experiment, we compare our hierarchical model to the equivalent IBP model. In each case, hyperparameters are specified with respect to the IBP (using cross-validation by evaluating the likelihood of a holdout set) and held fixed for the hierarchical factor model. Finally all hyperparameters of the hierarchical model that were not marginalized out were held constant over all experiments, in particular $c = 1$ and $\alpha_\nu = 1$.

### 5.1  Experiment I: Digits

In this experiment we took examples of images of hand-written digits from the MNIST dataset. Following [10], the dataset consisted of 1000 examples of images of the digit 3 where the handwritten digit images are first preprocessed by projecting onto the first 64 PCA components. To model MNIST digits, we augment both the IBP and the hierarchical model with a matrix $G$ of the same size as $Z$ and with i.i.d. zero mean and unit variance elements. Each data object, $x_n$ is modeled as: $x_n \mid Z, G, \theta, \sigma_x^2 \sim \mathcal{N}((z_{n,:} \odot g_{n,:})\theta, \sigma_X^2 I)$ where $\odot$ is the Hadamard (element-wise) product. The inclusion of $G$ introduces an additional step to our Gibbs sampling procedure, however the rest of the hierarchical infinity factor model is as described in Sect. 3. In order to assess the success of our hierarchical IFM in capturing higher-order factors present in the MNIST data, we consider a de-noising task. Random noise (std=0.5) was added to a post-processed test set and the models were evaluated in its ability to recover the noise-free version of a set of 500 examples not used in training. Fig. 2 (a) presents a comparison of the log likelihood of the (noise-free) test-set for both the hierarchical model and the IBP model. The figure shows that the 2-layer noisy-or model gives significantly more likelihood to the pre-corrupted data than the IBP, indicating that the noisy-or model was able to learn useful higher-order structure from MNIST data. One of the potential benefits of the style of model we propose here is that there is the opportunity for latent factors at one layer to share features at a lower layer. Fig. 2 illustrates the conditional mode of the random weight matrix $V$ (conditional on a sample of the other variables) and shows that there is significant sharing of low-level features by the higher-layer factors. Fig. 2 (d)-(e) compare the features (sampled rows of the $\theta$ matrix) learned by both the IBP and by the hierarchical noisy-or factor model. Interestingly, the sampled features learned in the hierarchical model appear to be slightly more spatially localized and sparse. Fig. 2 (f)-(i) illustrates some of the marginals that arise from the Gibbs sampling inference process. Interestingly, the IBP model infers a greater number of latent factors that did the 2-layer

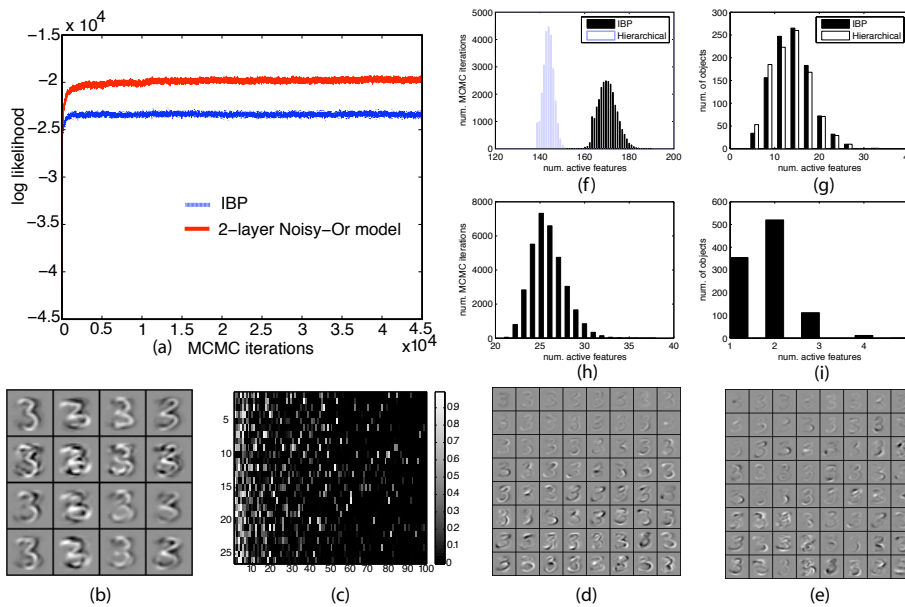

Figure 2: (a) The log likelihood of a de-noised testset. Corrupted (with 0.5-std Gaussian noise) versions of test examples were provided to the factor models and the likelihood of the noise-free testset was evaluated for both an IBP-based model as well as for the 2-layer noisy-or model. The two layer model shown substantial improvement in log likelihood. (b) Reconstruction of noisy examples. The top row shows the original values for a collection of digits. The second row shows their corrupted versions; while the third and fourth row show the reconstructions for the IBP-based model and the 2 layer noisy-or respectively. (c) A subset of the $V$ matrix. The rows of $V$ are indexed by $j$ while the columns of $V$ are indexed by $k$. The vertical striping pattern is evidence of significant sharing of lower-layer features among the upper-layer factors. (d)-(e) The most frequent 64 features (rows of the $\theta$ matrix) for (d) the IBP and for (e) the 2-layer infinite noisy-or factor model. (f) A comparison of the distributions of the number of active elements between the IBP and the noisy-or model. (g) A comparison of the number of active (lower-layer) factors possessed by an object between the IBP and the hierarchical model. (h) the distribution of upper-layer active factors and (i) the number of active factors found in an object.

noisy-or model (at the first layer). However, the distribution over factors active for each data object is nearly identical. This suggests the possibility that the IBP is maintaining specialized factors that possibly represent a superposition of frequently co-occurring factors that the noisy-or model has captured more compactly.

## 5.2 Experiment II: Music Tags

Returning to our motivating example from the introduction, we extracted tags and tag frequencies from the social music website Last.fm using the Audioscrobbler web service. The data is in the form of counts[1] of tag assignment for each artist. Our goal in modeling this data is to reduce this often noisy collection of tags to a sparse representation for each artist. We will adopt a different approach to the standard Latent Dirichlet Allocation (LDA) document processing strategy of modeling the document – or in this case tag collection – as having been generated from a mixture of tag multinomials. We wish to distinguish between an artist that everyone agrees is both country and rock versus an artist that people are divided whether they are rock or country.

To this end, we can again make use of the conjugate noisy-or model to model the count data in the form of binomial probabilities, i.e. to the model defined in Sect. 3, we add the random weights $W_{kt} \overset{i.i.d}{\sim} \text{Beta}(a, b), \forall k.t$ connecting $Z$ to the data $X$ via the distribution: $X_{nt} \sim \text{Binomial}(1 - \prod_k (1 - z_{nk}W), C)$ where $C$ is the limit on the number of possible counts achievable. This would correspond to the number of people who ever contributed a tag to that artist. In the case of the Last.fm data C = 100. Maintaining conjugacy over W will require us to add an assignment parameter

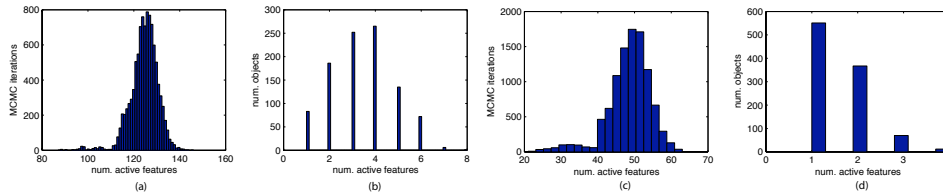

Figure 3: The distribution of active features for the noisy-or model at the (a) lower-layer and (c) the upper-layer. The distribution over active features per data object for the (b) upper-layer and (d) lower-layer.

$B_{nt}$ whose role is analogous to $A_{nk}$. With the model thus specified, we present a dataset of 1000 artists with a vocabulary size of 100 tags representing a total of 312134 counts. Fig. 3 shows the result running the Gibbs sampler for 10000 iterations. As the figure shows, both layers are quite sparse. Generally, most of the features learned in the first layer are dominated by one to three tags. Most features at the second layer cover a broader range of tags. The two most probable factors to emerge at the upper layer are associated with the tags (in order of probability):

1. `electronic, electronica, chillout, ambient, experimental`
2. `pop, rock, 80s, dance, 90s`

The ability of the 2-layer noisy-or model to capture higher-order structure in the tag data was again assessed though a comparison to the standard IBP using the noisy-or observation model above. The model was also compared against a more standard latent factor model with the latent representation $\eta_{nk}$ modeling the data through a generalized linear model: $X_{nt} \sim \text{Binomial}(\text{Logistic}(\eta_{n,:}O_{:,t}), C)$, where the function $\text{Logistic}(.)$ is the logistic sigmoid link function and the latent representation $\eta_{nk} \sim \mathcal{N}(0, \Sigma_\eta)$ are normally distributed. In this case, inference is performed via a Metropolis-Hastings MCMC method that mixes readily. The test data was missing 90% of the tags and the models were evaluated by their success in imputing the missing data from the 10% that remained. Here again, the 2-Layer Noisy-Or model achieved superior performance, as measured by the marginal log likelihood on a hold out set of 600 artist-tag collections. Interestingly both sparse models – the IBP and the noisy-or model – dramatically out performed the generalized latent linear model.

| Method | NLL |
|---:|---|
| Gen. latent linear model (Best Dim = 30) | 8.7781e05 $\pm$ 0.02e05 |
| IBP | 5.638e05 $\pm$ 0.001e05 |
| 2-Layer Noisy-Or IFM | 5.542e05 $\pm$ 0.001e05 |

## 6 Discussion

We have defined a noisy-or mechanism that allows one infinite factor model to act as a prior for another infinite factor model. The model permits high-order structure to be captured in a factor model framework while maintaining an efficient sampling algorithm. The model presented here is similar in spirit to the hierarchical Beta process, [11] in the sense that both models define a hierarchy of unbounded latent factor models. However, while the hierarchical Beta process can be seen as a way to group objects in the data-set with similar features, our model provides a way to group features that frequently co-occur in the data-set. It is perhaps more similar in spirit to the work of [9] who also sought a means of associating latent factors in an IBP, however their work does not act directly on the unbounded binary factors as ours does. Recently the question of how to define a hierarchical factor model to induce correlations between lower-layer factors was addressed by [3] with their IBP-IBP model. However, unlike our model, where the dependencies induced by the upper-layer factors via an noisy-or mechanism, the IBP-IBP model models correlations via an *AND* construct through the interaction of binary factors.

**Acknowledgments**

The authors acknowledge the support of NSERC and the Canada Research Chairs program. We also thank Last.fm for making the tag data publicly available and Paul Lamere for his help in processing the tag data.

## Footnotes

[1]The publicly available data is normalized to maximum value 100.

# References

[1] Robert J. Connor and James E. Mosimann. Concepts of independence for proportions with a generalization of the Dirichlet distribution. *Journal of the American Statistical Association*, 64(325):194–206, 1969.

[2] Aaron C. Courvile, Douglas Eck, and Yoshua Bengio. An infinite factor model hierarchy via a noisy-or mechanism: Supplemental material. Supplement to the NIPS paper.

[3] Finale Doshi-Velez and Zoubin Ghahramni. Correlated nonparametric latent feature models. In *Proceedings of the 25 th Conference on Uncertainty in Artificial Intelligence*, 2009.

[4] W. R. Gilks and P. Wild. Adaptive rejection sampling for Gibbs sampling. *Applied Statistics*, 41(2):337–348, 1992.

[5] Tom Griffiths and Zoubin Ghahramani. Infinite latent feature models and the indian buffet process. In *Advances in Neural Information Processing Systems 18*, Cambridge, MA, 2006. MIT Press.

[6] Max Henrion. Practical issues in constructing a bayes' belief network. In *Proceedings of the Proceedings of the Third Conference Annual Conference on Uncertainty in Artificial Intelligence (UAI-87)*, page 132?139, New York, NY, 1987. Elsevier Science.

[7] Hemant Ishwaran and Lancelot F. James. Gibbs sampling methods for stick-breaking priors. *American Statistical Association*, 96(453):161–173, 2001.

[8] Michael Kearns and Yishay Mansour. Exact inference of hidden structure from sample data in noisy-or networks. In *Proceedings of the 14 th Conference on Uncertainty in Artificial Intelligence*, pages 304–310, 1998.

[9] Piyush Rai and Hal Daumé III. The infinite hierarchical factor regression model. In Daphne Koller, Dale Schuurmans, Yoshua Bengio, and Léon Bottou, editors, *Advances in Neural Information Processing Systems 21*, 2009.

[10] Yee Whye Teh, Dilan Görür, and Zoubin Ghahramani. Stick-breaking construction for the indian buffet process. In *Proceedings of the Eleventh International Conference on Artifical Intelligence and Statistics (AISTAT 2007).*, 2007.

[11] Romain Thibaux and Michael I. Jordan. Hierarchical beta process and the indian buffet process. In *Proceedings of the Eleventh International Conference on Artifical Intelligence and Statistics (AISTAT 2007).*, 2007.

